# Invariant Feature Extraction and Classification in Kernel Spaces

Sebastian Mika[1], Gunnar Rätsch[1], Jason Weston[2],
Bernhard Schölkopf[3], Alex Smola[4], and Klaus-Robert Müller[1]

1 GMD FIRST, Kekulèstr. 7, 12489 Berlin, Germany
2 Barnhill BioInformatics, 6709 Waters Av., Savannah, GR 31406, USA
3 Microsoft Research Ltd., 1 Guildhall Street, Cambridge CB2 3NH, UK
4 Australian National University, Canberra, 0200 ACT, Australia

{mika, raetsch, klaus}@first.gmd.de, jasonw@dcs.rhbnc.ac.uk
bsc@microsoft.com, Alex.Smola.anu.edu.au

## Abstract

We incorporate prior knowledge to construct nonlinear algorithms
for invariant feature extraction and discrimination. Employing a
unified framework in terms of a nonlinear variant of the Rayleigh
coefficient, we propose non-linear generalizations of Fisher's dis-
criminant and oriented PCA using Support Vector kernel functions.
Extensive simulations show the utility of our approach.

## 1 Introduction

It is common practice to preprocess data by extracting linear or nonlinear features.
The most well-known feature extraction technique is principal component analysis
PCA (e.g. [3]). It aims to find an orthonormal, ordered basis such that the $i$-th
direction describes as much variance as possible while maintaining orthogonality to
all other directions. However, since PCA is a linear technique, it is too limited to
capture interesting nonlinear structure in a data set and nonlinear generalizations
have been proposed, among them Kernel PCA [14], which computes the principal
components of the data set mapped nonlinearly into some high dimensional feature
space $\mathcal{F}$.
Often one has prior information, for instance, we might know that the sample is
corrupted by noise or that there are invariances under which a classification should
not change. For feature extraction, the concepts of known noise or transformation
invariance are to a certain degree equivalent, i.e. they can both be interpreted as
causing a change in the feature which ought to be minimized. Clearly, invariance
alone is not a sufficient condition for a good feature, as we could simply take the
constant function. What one would like to obtain is a feature which is as invariant
as possible while still covering as much of the information necessary for describing
the particular data. Considering only one (linear) feature vector $w$ and restricting
to first and second order statistics of the data one arrives at a maximization of the
so called *Rayleigh* coefficient

$$J(w) = \frac{w^\top S_I w}{w^\top S_N w},\tag{1}$$

where $w$ is the feature vector and $S_I$, $S_N$ are matrices describing the desired and undesired properties of the feature, respectively (e.g. information and noise). If $S_I$ is the data covariance and $S_N$ the noise covariance, we obtain *oriented PCA* [3]. If we leave the field of data description to perform supervised classification, it is common to choose $S_I$ as the separability of class centers (between class variance) and $S_N$ to be the within class variance. In that case, we recover the well known Fisher Discriminant [7]. The ratio in (1) is maximized when we cover much of the information coded by $S_I$ while avoiding the one coded by $S_N$. The problem is known to be solved, in analogy to PCA, by a generalized symmetric eigenproblem $S_I w = \lambda S_N w$ [3], where $\lambda \in \mathbb{R}$ is the corresponding (biggest) eigenvalue.

In this paper we generalize this setting to a nonlinear one. In analogy to [8, 14] we first map the data via some nonlinear mapping $\Phi$ to some high-dimensional feature space $\mathcal{F}$ and then optimize (1) in $\mathcal{F}$. To avoid working with the mapped data explicitly (which might be impossible if $\mathcal{F}$ is infinite dimensional) we introduce support vector kernel functions [11], the well-known kernel trick. These kernel functions $k(x, y)$ compute a dot product in some feature space $\mathcal{F}$, i.e. $k(x, y) = (\Phi(x) \cdot \Phi(y))$. Formulating the algorithms in $\mathcal{F}$ using $\Phi$ only in dot products, we can replace any occurrence of a dot product by the kernel function $k$. Possible choices for $k$ which have proven useful e.g. in Support Vector Machines [2] or Kernel PCA [14] are Gaussian RBF, $k(x, y) = \exp(-\|x - y\|^2/c)$, or polynomial kernels, $k(x, y) = (x \cdot y)^d$, for some positive constants $c \in \mathbb{R}$ and $d \in \mathbb{N}$, respectively.

The remainder of this paper is organized as follows: The next section shows how to formulate the optimization problem induced by (1) in feature space. Section 3 considers various ways to find Fisher's Discriminant in $\mathcal{F}$; we conclude with extensive experiments in section 4 and a discussion of our findings.

## 2 Kernelizing the Rayleigh Coefficient

To optimize (1) in some kernel feature space $\mathcal{F}$ we need to find a formulation which uses only dot products of $\Phi$-images. As numerator and denominator are both scalars this can be done independently. Furthermore, the matrices $S_I$ and $S_N$ are basically covariances and thus the sum over outer products of $\Phi$-images. Therefore, and due to the linear nature of (1) every solution $w \in \mathcal{F}$ can be written as an expansion in terms of mapped training data[1], i.e.

$$w = \sum_{i=1}^{\ell} \alpha_i \Phi(x_i). \tag{2}$$

To define some common choices in $\mathcal{F}$ let $\mathcal{X} = \{x_1, \ldots, x_\ell\}$ be our training sample and, where appropriate, $\mathcal{X}_1 \cup \mathcal{X}_2 = \mathcal{X}, \mathcal{X}_1 \cap \mathcal{X}_2 = \emptyset$, two subclasses (with $|\mathcal{X}_i| = \ell_i$). We get the full covariance of $\mathcal{X}$ by

$$C = \frac{1}{\ell} \sum_{x \in \mathcal{X}} (\Phi(x) - m)(\Phi(x) - m)^\top \text{ with } m = \frac{1}{\ell} \sum_{x \in \mathcal{X}} \Phi(x), \tag{3}$$

$$
\begin{aligned}
\langle w, Sw \rangle &= \langle (v_1 + v_2), S(v_1 + v_2) \rangle \\
&= \langle (v_1 + v_2)S, v_1 \rangle \\
&= \langle v_1, Sv_1 \rangle
\end{aligned}
$$

As $v_1$ lies in the span of the $\Phi(x_i)$ and $S$ only operates on this subspace there exist an expansion of $w$ which maximizes $J(w)$.

which could be used as $S_I$ in oriented Kernel PCA. For $S_N$ we could use an estimate of the noise covariance, analogous to the definition of $C$ but over mapped patterns sampled from the assumed noise distribution. The standard formulation of the Fisher discriminant in $\mathcal{F}$, yielding the *Kernel Fisher Discriminant* (KFD) [8] is given by

$$S_W = \sum_{i=1,2} \sum_{x \in \mathcal{X}_i} (\Phi(x) - m_i)(\Phi(x) - m_i)^\top \quad \text{and} \quad S_B = (m_2 - m_1)(m_2 - m_1)^\top,$$

the within-class scatter $S_W$ (as $S_N$), and the between class scatter $S_B$ (as $S_I$). Here $m_i$ is the sample mean for patterns from class $i$.

To incorporate a known invariance e.g. in oriented Kernel PCA, one could use the tangent covariance matrix [12],

$$T = \frac{1}{\ell t^2} \sum_{x \in \mathcal{X}} (\Phi(x) - \Phi(\mathcal{L}_t x))(\Phi(x) - \Phi(\mathcal{L}_t x))^\top \text{ for some small } t > 0. \qquad (4)$$

Here $\mathcal{L}_t$ is a local 1-parameter transformation. $T$ is a finite difference approximation $t$ of the covariance of the tangent of $\mathcal{L}_t$ at point $\Phi(x)$ (details e.g. in [12]). Using $S_I = C$ and $S_N = T$ in oriented Kernel PCA, we impose invariance under the local transformation $\mathcal{L}_t$. Crucially, this matrix is not only constructed from the training patterns $\mathcal{X}$. Therefore, the argument used to find the expansion (2) is slightly incorrect. Nevertheless, we can assume that (2) is a reasonable approximation for describing the variance induced by $T$.

Multiplying either of these matrices from the left and right with the expansion (2), we can find a formulation which uses only dot products. For the sake of brevity, we only give the explicit formulation of (1) in $\mathcal{F}$ for KFD (cf. [8] for details). Defining $(\mu_i)_j = \frac{1}{\ell_i} \sum_{x \in \mathcal{X}_i} k(x_j, x)$ we can write (1) for KFD as

$$J(\alpha) = \frac{(\alpha^\top \mu)^2}{\alpha^\top N \alpha} = \frac{\alpha^\top M \alpha}{\alpha^\top N \alpha}, \qquad (5)$$

where $N = KK^\top - \sum_{i=1,2} \ell_i \mu_i \mu_i^\top$, $\mu = \mu_2 - \mu_1$, $M = \mu \mu^\top$, and $K_{ij} = k(x_i, x_j)$. The results for other choices of $S_I$ and $S_N$ in $\mathcal{F}$ as for the cases of oriented kernel PCA or transformation invariance can be obtained along the same lines. Note that we still have to maximize a Rayleigh coefficient. However, now it is a quotient in terms of expansion coefficients $\alpha$, and not in terms of $w \in \mathcal{F}$ which is a potentially infinite-dimensional space. Furthermore, it is well known that the solution for this special eigenproblem is in the direction of $N^{-1}(\mu_2 - \mu_1)$ [7], which can be solved using e.g. a Cholesky factorization of $N$. The projection of a new pattern $x$ onto $w$ in $\mathcal{F}$ can then be computed by

$$(w \cdot \Phi(x)) = \sum_{i=1}^{\ell} \alpha_i k(x_i, x). \qquad (6)$$

## 3   Algorithms

Estimating a covariance matrix with rank up to $\ell$ from $\ell$ samples is ill-posed. Furthermore, by performing an explicit centering in $\mathcal{F}$ each covariance matrix loses one more dimension, i.e. it has only rank $\ell - 1$ (even worse, for KFD the matrix $N$ has rank $\ell - 2$). Thus the ratio in (1) is not well defined anymore, as the denominator might become zero. In the following we will propose several ways to deal with this problem in KFD. Furthermore we will tackle the question how to solve the optimization problem of KFD more efficiently. So far, we have an eigenproblem of size $\ell \times \ell$. If $\ell$ becomes large this is numerically demanding. Reformulations of the original problem allow to overcome some of these limitations. Finally, we describe the connection between KFD and RBF networks.

## 3.1   Regularization and Solution on a Subspace

As noted before, the matrix $N$ has only rank $\ell - 2$. Besides numerical problems which can cause the matrix $N$ to be not even positive, we could think of imposing some regularization to control capacity in $\mathcal{F}$. To this end, we simply add a multiple of the identity matrix to $N$, i.e. replace $N$ by $N_\mu$ where

$$N_\mu := N + \mu I. \tag{7}$$

This can be viewed in different ways: (i) for $\mu > 0$ it makes the problem feasible and numerically more stable as $N_\mu$ becomes positive; (ii) it can be seen as decreasing the bias in sample based estimation of eigenvalues (cf. [6]); (iii) it imposes a regularization on $\|\alpha\|^2$, favoring solutions with small expansion coefficients. Furthermore, one could use other regularization type additives to $N$, e.g. penalizing $\|w\|^2$ in analogy to SVM (by adding the kernel matrix $K_{ij} = \mathrm{k}(x_i, x_j)$).
To optimize (5) we need to solve an $\ell \times \ell$ eigenproblem, which might be intractable for large $\ell$. As the solutions are not sparse one can not directly use efficient algorithms like chunking for Support Vector Machines (cf. [13]). To this end, we might restrict the solution to lie in a subspace, i.e. instead of expanding $w$ by (2) we write

$$w = \sum_{i=1}^{m} \alpha_i \Phi(z_i), \tag{8}$$

with $m < l$. The patterns $z_i$ could either be a subset of the training patterns $\mathcal{X}$ or e.g. be estimated by some clustering algorithm. The derivation of (5) does not change, only $K$ is now $m \times \ell$ and we end up with $m \times m$ matrices $N$ and $M$. Another advantage is, that it increases the rank of $N$ (relative to its size) although there still might be some need for regularization.

## 3.2   Quadratic optimization and Sparsification

Even if $N$ has full rank, maximizing (5) is underdetermined: if $\alpha$ is optimal, then so is any multiple thereof. Since $\alpha^\top M \alpha = (\alpha^\top \mu)^2$, $M$ has rank one. Thus we can seek for a vector $\alpha$, such that $\alpha^\top N \alpha$ is minimal for fixed $\alpha^\top \mu$ (e.g. to 1). The solution is unique and we can find the optimal $\alpha$ by solving the quadratic optimization problem:

$$\min \ \alpha^\top N \alpha \quad \text{subject to} \ \alpha^\top \mu = 1. \tag{9}$$

Although the quadratic optimization problem is not easier to solve than the eigenproblem, it has an appealing interpretation. The constraint $\alpha^\top \mu = 1$ ensures, that the average class distance, projected onto the direction of discrimination, is constant, while the intra class variance is minimized, i.e. we maximize the *average* margin. Contrarily, the SVM approach [2] optimizes for a large *minimal* margin. Considering (9) we are able to overcome another shortcoming of KFD. The solutions $\alpha$ are *not* sparse and thus evaluating (6) is expensive. To solve this we can add an $l_1$-regularizer $\lambda \|\alpha\|_1$ to the objective function, where $\lambda$ is a regularization parameter allowing us to adjust the degree of sparseness.

## 3.3   Connection to RBF Networks

Interestingly, there exists a close connection between RBF networks (e.g. [9, 1]) and KFD. If we add no regularization and expand in all training patterns, we find that an optimal $\alpha$ is given by $\alpha = K^{-1}y$, where $K$ is the symmetric, positive matrix of all kernel elements $\mathrm{k}(x_i, x_j)$ and $y$ the $\pm 1$ label vector[2]. A RBF-network with the

| | RBF | AB | $AB_R$ | SVM | KFD |
|---|---|---|---|---|---|
| Banana | **10.8±0.06** | 12.3±0.07 | *10.9±0.04* | 11.5±0.07 | **10.8±0.05** |
| B.Cancer | 27.6±0.47 | 30.4±0.47 | 26.5±0.45 | *26.0±0.47* | **25.8±0.46** |
| Diabetes | 24.3±0.19 | 26.5±0.23 | 23.8±0.18 | *23.5±0.17* | **23.2±0.16** |
| German | 24.7±0.24 | 27.5±0.25 | 24.3±0.21 | **23.6±0.21** | *23.7±0.22* |
| Heart | 17.6±0.33 | 20.3±0.34 | 16.5±0.35 | **16.0±0.33** | *16.1±0.34* |
| Image | 3.3±0.06 | **2.7±0.07** | **2.7±0.06** | *3.0±0.06* | 4.8±0.06 |
| Ringnorm | 1.7±0.02 | 1.9±0.03 | *1.6±0.01* | 1.7±0.01 | **1.5±0.01** |
| F.Sonar | 34.4±0.20 | 35.7±0.18 | 34.2±0.22 | **32.4±0.18** | *33.2±0.17* |
| Splice | *10.0±0.10* | 10.1±0.05 | **9.5±0.07** | 10.9±0.07 | 10.5±0.06 |
| Thyroid | 4.5±0.21 | *4.4±0.22* | 4.6±0.22 | 4.8±0.22 | **4.2±0.21** |
| Titanic | 23.3±0.13 | *22.6±0.12* | *22.6±0.12* | **22.4±0.10** | 23.2±0.20 |
| Twonorm | 2.9±0.03 | 3.0±0.03 | *2.7±0.02* | 3.0±0.02 | **2.6±0.02** |
| Waveform | 10.7±0.11 | 10.8±0.06 | **9.8±0.08** | *9.9±0.04* | *9.9±0.04* |

Table 1: Comparison between KFD, single RBF classifier, AdaBoost (AB), regul. AdaBoost $(AB_R)$ and SVMs (see text). Best result in bold face, second best in italics.

same kernel at each sample and fixed kernel width gives the same solution, if the mean squared error between labels and output is minimized. Also for the case of restricted expansions (8) there exists a connection to RBF networks with a smaller number of centers (cf. [4]).

## 4    Experiments

**Kernel Fisher Discriminant**    Figure 1 shows an illustrative comparison of the features found by KFD, and Kernel PCA. The KFD feature discriminates the two classes, the first Kernel PCA feature picks up the important nonlinear structure.

To evaluate the performance of the KFD on real data sets we performed an extensive comparison to other state-of-the-art classifiers, whose details are reported in [8].[3] We compared the Kernel Fisher Discriminant and Support Vector Machines, both with Gaussian kernel, to AdaBoost [5], and regularized AdaBoost [10] (cf. table 1). For KFD we used the regularized within-class scatter (7) and computed projections onto the optimal direction $w \in \mathcal{F}$ by means of (6). To use $w$ for classification we have to estimate a threshold. This can be done by e.g. trying all thresholds between two outputs on the training set and selecting the median of those with the smallest empirical error, or (as we did here) by computing the threshold which maximizes the margin on the outputs in analogy to a Support Vector Machine, where we deal with errors on the trainig set by using the SVM soft margin approach. A disadvantage of this is, however, that we have to control the regularization constant for the slack variables.    The results in table 1 show the average test error and the standard

---

If $K$ has full rank, the null space of $D$, which is spanned by $y_1$ and $y_2$, is the null space of $N$. For $\bar{\alpha} = K^{-1}y$ we get $\bar{\alpha}^T N \bar{\alpha} = 0$ and $\bar{\alpha}^T \mu \neq 0$. As we are free to fix the constraint $\alpha^T \mu$ to any positive constant (not just 1), $\bar{\alpha}$ is also feasible.

[3]The breast cancer domain was obtained from the University Medical Center, Inst. of Oncology, Ljubljana, Yugoslavia. Thanks to M. Zwitter and M. Soklic for the data. All data sets used in the experiments can be obtained via http://www.first.gmd.de/~raetsch/.

Figure 1: Comparison of feature found by KFD (left) and first Kernel PCA feature (right). Depicted are two classes (information only used by KFD) as dots and crosses and levels of same feature value. Both with polynomial kernel of degree two, KFD with the regularized within class scatter (7) ($\mu = 10^{-3}$).

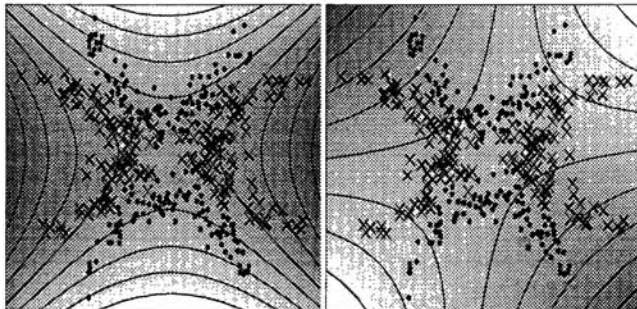

deviation of the averages' estimation, over 100 runs with different realizations of the datasets. To estimate the necessary parameters, we ran 5-fold cross validation on the first five realizations of the training sets and took the model parameters to be the median over the five estimates (see [10] for details of the experimental setup).

**Using prior knowledge.** A toy example (figure 2) shows a comparison of Kernel PCA and oriented Kernel PCA, which used $S_I$ as the full covariance (3) and as noise matrix $S_N$ the tangent covariance (4) of (i) rotated patterns and (ii) along the x-axis translated patterns. The toy example shows how imposing the desired invariance yields meaningful invariant features.

In another experiment we incorporated prior knowledge in KFD. We used the USPS database of handwritten digits, which consists of 7291 training and 2007 test patterns, each 256 dimensional gray scale images of the digits $0 \ldots 9$. We used the regularized within-class scatter (7) ($\mu = 10^{-3}$) as $S_N$ and added to it an multiple $\lambda$ of the tangent covariance (4), i.e. $S_N = N_\mu + \lambda T$. As invariance transformations we have chosen horizontal and vertical translation, rotation, and thickening (cf. [12]), where we simply averaged the matrices corresponding to each transformation. The feature was extracted by using the restricted expansion (8), where the patterns $z_i$ were the first 3000 training samples. As kernel we have chosen a Gaussian of width $0.3 \cdot 256$, which is optimal for SVMs [12]. For each class we trained one KFD which classified this class against the rest and computed the 10-class error by the winner-takes-all scheme. The threshold was estimated by minimizing the empirical risk on the normalized outputs of KFD.

Without invariances, i.e. $\lambda = 0$, we achieved a test error of 3.7%, slightly better than a plain SVM with the same kernel (4.2%) [12]. For $\lambda = 10^{-3}$, using the tangent covariance matrix led to a very slight improvement to 3.6%. That the result was not significantly better than the corresponding one for KFD (3.7%) can be attributed to the fact that we used the same expansion coefficients in both cases. The tangent covariance matrix, however, lives in a slightly different subspace. And indeed, a subsequent experiment where we used vectors which were obtained by clustering a larger dataset, including virtual examples generated by the appropriate invariance transformation, led to 3.1%, comparable to an SVM using prior knowledge (e.g. [12]; best SVM result 2.9% with local kernel and virtual support vectors).

## 5 Conclusion

In the task of learning from data it is equivalent to have prior knowledge about e.g. invariances or about specific sources of noise. In the case of feature extraction, we seek features which are sufficiently (noise-) invariant while still describing interesting structure. Oriented PCA and, closely related, Fisher's Discriminant, use particularly simple features, since they only consider first and second order statistics for maximizing the Rayleigh coefficient (1). Since linear methods can be too restricted in many real-world applications, we used Support Vector Kernel functions to obtain nonlinear versions of these algorithms, namely oriented Kernel PCA and Kernel Fisher Discriminant analysis.

Our experiments show that the Kernel Fisher Discriminant is competitive or in

Figure 2: Comparison of first features found by Kernel PCA and oriented Kernel PCA (see text); from left to right: KPCA, OKPCA with rotation and translation invariance; all with Gaussian kernel.

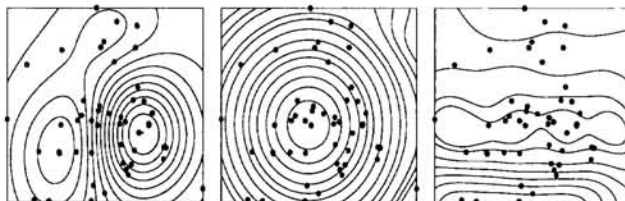

some cases even superior to the other state-of-the-art algorithms tested. Interestingly, both SVM and KFD construct a hyperplane in $\mathcal{F}$ which is in some sense optimal. In many cases, the one given by the solution $w$ of KFD is superior to the one of SVMs. Encouraged by the preliminary results for digit recognition, we believe that the reported results can be improved, by incorporating different invariances and using e.g. local kernels [12].

Future research will focus on further improvements on the algorithmic complexity of our new algorithms, which is so far larger than the one of the SVM algorithm, and on the connection between KFD and Support Vector Machines (cf. [16, 15]).

**Acknowledgments**   This work was partially supported by grants of the DFG (JA 379/5-2,7-1,9-1) and the EC STORM project number 25387 and carried out while BS and AS were with GMD First.

# References

[1] C.M. Bishop. *Neural Networks for Pattern Recognition*. Oxford Univ. Press, 1995.

[2] B. Boser, I. Guyon, and V.N. Vapnik. A training algorithm for optimal margin classifiers. In D. Haussler, editor, *Proc. COLT*, pages 144–152. ACM Press, 1992.

[3] K.I. Diamantaras and S.Y. Kung. *Principal Component Neural Networks*. Wiley, New York, 1996.

[4] B.Q. Fang and A.P. Dawid. Comparison of full bayes and bayes-least squares criteria for normal discrimination. *Chinese Journal of Applied Probability and Statistics*, 12:401–410, 1996.

[5] Y. Freund and R.E. Schapire. A decision-theoretic generalization of on-line learning and an application to boosting. In *EuroCOLT 94*. LNCS, 1994.

[6] J.H. Friedman. Regularized discriminant analysis. *Journal of the American Statistical Association*, 84(405):165–175, 1989.

[7] K. Fukunaga. *Introduction to Statistical Pattern Recognition*. Academic Press, San Diego, 2nd edition, 1990.

[8] S. Mika, G. Rätsch, J. Weston, B. Schölkopf, and K.-R. Müller. Fisher discriminant analysis with kernels. In Y.-H. Hu, J. Larsen, E. Wilson, and S. Douglas, editors, *Neural Networks for Signal Processing IX*, pages 41–48. IEEE, 1999.

[9] J. Moody and C. Darken. Fast learning in networks of locally-tuned processing units. *Neural Computation*, 1(2):281–294, 1989.

[10] G. Rätsch, T. Onoda, and K.-R. Müller. Soft margins for adaboost. Technical Report NC-TR-1998-021, Royal Holloway College, University of London, UK, 1998.

[11] S. Saitoh. *Theory of Reproducing Kernels and its Applications*. Longman Scientific & Technical, Harlow, England, 1988.

[12] B. Schölkopf. *Support vector learning*. Oldenbourg Verlag, 1997.

[13] B. Schölkopf, C.J.C. Burges, and A.J. Smola, editors. *Advances in Kernel Methods – Support Vector Learning*. MIT Press, 1999.

[14] B. Schölkopf, A.J. Smola, and K.-R. Müller. Nonlinear component analysis as a kernel eigenvalue problem. *Neural Computation*, 10:1299–1319, 1998.

[15] A. Shashua. On the relationship between the support vector machine for classification and sparsified fisher's linear discriminant. *Neural Processing Letters*, 9(2):129–139, April 1999.

[16] S. Tong and D. Koller. Bayes optimal hyperplanes $\rightarrow$ maximal margin hyperplanes. Submitted to IJCAI'99 Workshop on Support Vector Machines (`robotics.stanford.edu/~koller/`), 1999.
